# Heavy-Tailed Symmetric Stochastic Neighbor Embedding

**Zhirong Yang**
The Chinese University of Hong Kong
Helsinki University of Technology
zhirong.yang@tkk.fi

**Irwin King**
The Chinese University of Hong Kong
king@cse.cuhk.edu.hk

**Zenglin Xu**
The Chinese University of Hong Kong
Saarland University & MPI for Informatics
zlxu@cse.cuhk.edu.hk

**Erkki Oja**
Helsinki University of Technology
erkki.oja@tkk.fi

## Abstract

Stochastic Neighbor Embedding (SNE) has shown to be quite promising for data visualization. Currently, the most popular implementation, t-SNE, is restricted to a particular Student t-distribution as its embedding distribution. Moreover, it uses a gradient descent algorithm that may require users to tune parameters such as the learning step size, momentum, etc., in finding its optimum. In this paper, we propose the Heavy-tailed Symmetric Stochastic Neighbor Embedding (HSSNE) method, which is a generalization of the t-SNE to accommodate various heavy-tailed embedding similarity functions. With this generalization, we are presented with two difficulties. The first is how to select the best embedding similarity among all heavy-tailed functions and the second is how to optimize the objective function once the heavy-tailed function has been selected. Our contributions then are: (1) we point out that various heavy-tailed embedding similarities can be characterized by their negative score functions. Based on this finding, we present a parameterized subset of similarity functions for choosing the best tail-heaviness for HSSNE; (2) we present a fixed-point optimization algorithm that can be applied to all heavy-tailed functions and does not require the user to set any parameters; and (3) we present two empirical studies, one for unsupervised visualization showing that our optimization algorithm runs as fast and as good as the best known t-SNE implementation and the other for semi-supervised visualization showing quantitative superiority using the homogeneity measure as well as qualitative advantage in cluster separation over t-SNE.

## 1 Introduction

Visualization as an important tool for exploratory data analysis has attracted much research effort in recent years. A multitude of visualization approaches, especially the nonlinear dimensionality reduction techniques such as Isomap [9], Laplacian Eigenmaps [1], *Stochastic Neighbor Embedding* (SNE) [6], manifold sculpting [5], and kernel maps with a reference point [8], have been proposed. Although they are reported with good performance on tasks such as unfolding an artificial manifold, they are often not successful at visualizing real-world data with high dimensionalities.

A common problem of the above methods is that most mapped data points are crowded together in the center without distinguished gaps that isolate data clusters. It was recently pointed out by van der Maaten and Hinton [10] that the "crowding problem" can be alleviated by using a heavy-tailed

distribution in the low-dimensional space. Their method, called *t-Distributed Stochastic Neighbor Embedding* (t-SNE), is adapted from SNE with two major changes: (1) it uses a symmetrized cost function; and (2) it employs a Student t-distribution with a single degree of freedom ($T_1$). In this way, t-SNE can achieve remarkable superiority in the discovery of clustering structure in high-dimensional data.

The t-SNE development procedure in [10] is restricted to the $T_1$ distribution as its embedding similarity. However, different data sets or other purposes of dimensionality reduction may require generalizing t-SNE to other heavy-tailed functions. The original t-SNE derivation provides little information for users on how to select the best embedding similarity among all heavy-tailed functions.

Furthermore, the original t-SNE optimization algorithm is not convenient when the symmetric SNE is generalized to use various heavy-tailed embedding similarity functions since it builds on the gradient descent approach with momenta. As a result, several optimization parameters need to be manually specified. The performance of the t-SNE algorithm depends on laborious selection of the optimization parameters. For instance, a large learning step size might cause the algorithm to diverge, while a conservative one might lead to slow convergence or poor annealed results. Although comprehensive strategies have been used to improve the optimization performance, they might be still problematic when extended to other applications or embedding similarity functions.

In this paper we generalize t-SNE to accommodate various heavy-tailed functions with two major contributions: (1) we propose to characterize heavy-tailed embedding similarities in symmetric SNE by their negative score functions. This further leads to a parameterized subset facilitating the choice of the best tail-heaviness; and (2) we present a general algorithm for optimizing the symmetric SNE objective with any heavy-tailed embedding similarities.

The paper is organized as follows. First we briefly review the related work of SSNE and t-SNE in Section 2. In Section 3, we present the generalization of t-SNE to our Heavy-tailed Symmetric SNE (HSSNE) method. Next, a fixed-point optimization algorithm for HSSNE is provided and its convergence is discussed in Section 4. In Section 5, we relate the EM-like behavior of the fixed-point algorithm to a pairwise local mixture model for an in-depth analysis of HSSNE. Section 6 presents two sets of experiments, one for unsupervised and the other for semi-supervised visualization. Finally, conclusions are drawn in Section 7.

## 2 Symmetric Stochastic Neighbor Embedding

Suppose the pairwise similarities of a set of $m$-dimensional data points $\mathcal{X} = \{x_i\}_{i=1}^n$ are encoded in a symmetric matrix $P \in \mathbb{R}_+^{n \times n}$, where $P_{ii} = 0$ and $\sum_{ij} P_{ij} = 1$. *Symmetric Stochastic Neighbor Embedding* (SSNE) [4, 10] seeks $r$-dimensional ($r \ll m$) representations of $\mathcal{X}$, denoted by $\mathcal{Y} = \{y_i\}_{i=1}^n$, such that

$$\mathcal{J}(Y) = D_{\text{KL}}(P||Q) = \sum_{i \neq j} P_{ij} \log \frac{P_{ij}}{Q_{ij}} \tag{1}$$

is minimized, where $Q_{ij} = q_{ij} / \sum_{a \neq b} q_{ab}$ are the normalized similarities in low-dimensional embedding and

$$q_{ij} = \exp\left(-\|y_i - y_j\|^2\right), \quad q_{ii} = 0. \tag{2}$$

The optimization of SSNE uses the gradient descent method with

$$\frac{\partial \mathcal{J}}{\partial y_i} = 4 \sum_j (P_{ij} - Q_{ij})(y_i - y_j). \tag{3}$$

A momentum term is added to the gradient in order to speed up the optimization:

$$Y^{(t+1)} = Y^{(t)} + \eta \frac{\partial \mathcal{J}}{\partial Y}\Big|_{Y=Y^{(t)}} + \beta(t)\left(Y^{(t)} - Y^{(t-1)}\right), \tag{4}$$

where $Y^{(t)} = [y_1^{(t)} \ldots y_n^{(t)}] \in \mathbb{R}^{r \times n}$ is the solution in matrix form at iteration $t$; $\eta$ is the learning rate; and $\beta(t)$ is the momentum amount at iteration $t$. Compared with an earlier method *Stochastic Neighbor Embedding* (SNE) [6], SSNE uses a symmetrized cost function with simpler gradients.

Most mapped points in the SSNE visualizations are often compressed near the center of the visualizing map without clear gaps that separate clusters of the data. The *t-Distributed Stochastic Neighbor*

*Embedding* (t-SNE) [10] addresses this crowding problem by using the Student t-distribution with a single degree of freedom

$$q_{ij} = (1 + \|y_i - y_j\|^2)^{-1}, \quad q_{ii} = 0, \tag{5}$$

as the embedding similarity distribution, which has a heavier tail than the Gaussian used in SNE and SSNE. For brevity we denote such distribution by $T_1$. Using this distribution yields the gradient of t-SNE:

$$\frac{\partial \mathcal{J}}{\partial y_i} = 4 \sum_j (P_{ij} - Q_{ij})(y_i - y_j)(1 + \|y_i - y_j\|^2)^{-1}. \tag{6}$$

In addition, t-SNE employs a number of strategies to overcome the difficulties in the optimization based on gradient descent.

## 3 Heavy-tailed SNE characterized by negative score functions

As the gradient derivation in [10] is restricted to the $T_1$ distribution, we derive the gradient with a general function that converts squared distances to similarities, with $T_1$ as a special case. In addition, the direct chain rule used in [10] may cause notational clutter and conceal the working components in the gradients. We instead employ the Lagrangian technique to simplify the derivation. Our approach can provide more insights of the working factor brought by the heavy-tailed functions.

Minimizing $\mathcal{J}(Y)$ in Equation (1) with respect to $Y$ is equivalent to the optimization problem:

$$\underset{q,Y}{\text{maximize}} \ \mathcal{L}(q, Y) = \sum_{ij} P_{ij} \log \frac{q_{ij}}{\sum_{a \neq b} q_{ab}} \tag{7}$$

$$\text{subject to} \ q_{ij} = H(\|y_i - y_j\|^2), \tag{8}$$

where the *embedding similarity function* $H(\tau) \geq 0$ can be any function that is monotonically decreasing with respect to $\tau$ for $\tau > 0$. Note that $H$ is not required to be defined as a probability function because the symmetric SNE objective already involves normalization over all data pairs.

The extended objective using the Lagrangian technique is given by

$$\tilde{\mathcal{L}}(q, Y) = \sum_{ij} P_{ij} \log \frac{q_{ij}}{\sum_{a \neq b} q_{ab}} + \sum_{ij} \lambda_{ij} \left[ q_{ij} - H(\|y_i - y_j\|^2) \right]. \tag{9}$$

Setting $\partial \tilde{\mathcal{L}}(q, Y)/\partial q_{ij} = 0$ yields $\lambda_{ij} = 1/\sum_{a \neq b} q_{ab} - P_{ij}/q_{ij}$. Inserting these Lagrangian multipliers to the gradient with respect to $y_i$, we have

$$\frac{\partial \mathcal{J}(Y)}{\partial y_i} = -\frac{\partial \tilde{\mathcal{L}}(q, Y)}{\partial y_i} = 4 \sum_j \left( \frac{1}{\sum_{a \neq b} q_{ab}} - \frac{P_{ij}}{q_{ij}} \right) \cdot q_{ij} \cdot \left( -\frac{h(\|y_i - y_j\|^2)}{q_{ij}} \right) (y_i - y_j) \tag{10}$$

$$= 4 \sum_j (P_{ij} - Q_{ij}) S(\|y_i - y_j\|^2)(y_i - y_j), \tag{11}$$

where $h(\tau) = dH(\tau)/d\tau$ and

$$S(\tau) = -\frac{d \log H(\tau)}{d\tau} \tag{12}$$

is the negative score function of $H$. For notational simplicity, we also write $S_{ij} = S(\|y_i - y_j\|^2)$.

We propose to characterize the tail heaviness of the similarity function $H$, relative to the one that leads to the Gaussian, by its negative score function $S$, also called *tail-heaviness function* in this paper. In this characterization, there is a functional operator $\mathcal{S}$ that maps every similarity function to a tail-heaviness function. For the baseline Gaussian similarity, $H(\tau) = \exp(-\tau)$, we have $\mathcal{S}(H) = 1$, i.e. $\mathcal{S}(H)(\tau) = 1$ for all $\tau$. As for the Student t-distribution of a single degree of freedom, $H(\tau) = (1 + \tau)^{-1}$ and thus $\mathcal{S}(H) = H$.

The above observation inspires us to further parameterize a family of tail-heaviness functions by the power of $H$: $\mathcal{S}(H, \alpha) = H^\alpha$ for $\alpha \geq 0$, where a larger $\alpha$ value corresponds to a heavier-tailed embedding similarity function. Such a function $H$ can be determined by solving the first-order differential equation $-d \log H(\tau)/d\tau = [H(\tau)]^\alpha$, which gives

$$H(\tau) = (\alpha\tau + c)^{-1/\alpha} \tag{13}$$

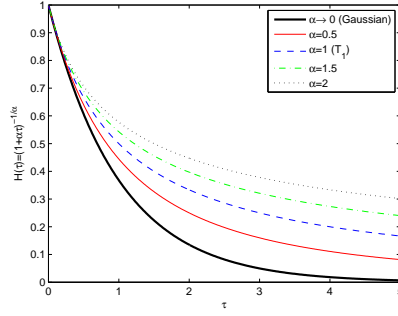

Figure 1: Several functions in the power family.

with $c$ a constant. Here we set $c = 1$ for a consistent generalization of SNE and t-SNE. Thus the Gaussian embedding similarity function, i.e. $H(\tau) = \exp(-\tau)$, is achieved when $\alpha \to 0$. Figure 1 shows a number of functions in the power family.

## 4   A fixed-Point optimization algorithm

Unlike many other dimensionality reduction approaches that can be solved by eigendecomposition in a single step, SNE and its variants require iterative optimization methods. Substantial efforts have been devoted to improve the efficiency and robustness of t-SNE optimization. However it remains unknown whether such a comprehensive implementation also works for other types of embedding similarity functions. Manually adjusting the involved parameters such as the learning rate and the momentum for every function is rather time-consuming and infeasible in practice.

Here we propose to optimize symmetric SNE by a fixed-point algorithm. After rearranging the terms in $\partial \mathcal{J} / \partial y_i = 0$ (see Equation (11)), we obtain the following update rule:

$$Y_{ki}^{(t+1)} = \frac{Y_{ki}^{(t)} \sum_j B_{ij} + \sum_j (A_{ij} - B_{ij}) Y_{kj}^{(t)}}{\sum_j A_{ij}}, \tag{14}$$

where $A_{ij} = P_{ij} S(\|y_i^{(t)} - y_j^{(t)}\|^2)$ and $B_{ij} = Q_{ij} S(\|y_i^{(t)} - y_j^{(t)}\|^2)$. Our optimization algorithm for HSSNE simply involves the iterative application of Equation (14). Compared with the original t-SNE optimization algorithm, our method requires no user-provided parameters such as the learning step size and momentum, which is more convenient for applications. The fixed-point algorithm usually converges, with the result satisfying the stationary condition $\partial \mathcal{J} / \partial Y = 0$. However, it is known that the update rule (14) can diverge in some cases, for example, when $Y_{ki}$ are large. Therefore, a proof without extra conditions cannot be constructed. Here we provide two approximative theoretical justifications for the algorithm.

Denote $\Delta = Y - Y^{(t)}$ and $\nabla$ the gradient of $\mathcal{J}$ with respect to $Y$. Let us first approximate the HSSNE objective by the first-order Taylor expansion at the current estimate $Y^{(t)}$:

$$\mathcal{J}(Y) \approx \mathcal{J}_{\text{lin}}(Y) = \mathcal{J}(Y^{(t)}) + \sum_{ki} \Delta_{ki} \nabla_{ki}^{(t)}. \tag{15}$$

Then we can construct an upper bound of $\mathcal{J}_{\text{lin}}(Y)$:

$$G(Y, Y^{(t)}) = \mathcal{J}_{\text{lin}}(Y) + \frac{1}{2} \sum_{ki} \Delta_{ki}^2 \sum_a A_{ia} \tag{16}$$

as $P_{ia}$ and $S_{ia}$ are all nonnegative. The bound is tight at $Y = Y^{(t)}$, i.e. $G(Y^{(t)}, Y^{(t)}) = \mathcal{J}_{\text{lin}}(Y^{(t)})$. Equating $\partial G(Y, Y^{(t)}) / \partial Y = 0$ implements minimization of $G(Y, Y^{(t)})$ and yields the update rule (14). Iteratively applying the update rule (14) thus results in a monotonically decreasing sequence of the linear approximation of HSSNE objective: $\mathcal{J}_{\text{lin}}(Y^{(t)}) \geq G(Y^{(t+1)}, Y^{(t)}) \geq \mathcal{J}_{\text{lin}}(Y^{(t+1)})$.

Even if the second-order terms in the Taylor expansion of $\mathcal{J}(Y)$ are also considered, the update rule (14) is still justified if $Y_{ki}$ or $Y_{ki}^{(t+1)} - Y_{ki}^{(t)}$ are small. Let $D^A$ and $D^B$ be diagonal matrices with $D_{ii}^A = \sum_j A_{ij}$ and $D_{ii}^B = \sum_j B_{ij}$. We can write $\mathcal{J}(Y) = \mathcal{J}_{\text{quad}}(Y) + O(\Delta^3)$, where

$$\mathcal{J}_{\text{quad}}(Y) = \mathcal{J}_{\text{lin}}(Y) + \frac{1}{2} \sum_{ijkl} \Delta_{ki} \Delta_{lj} H_{ijkl}. \tag{17}$$

With the approximated Hessian $H_{ijkl} = \delta_{kl} \left[ (D^A - A) - (D^B - B) \right]_{ij}$, the updating term $U_{ki}$ in Newton's method $Y_{ki}^{(t)} = Y_{ki}^{(t-1)} - U_{ki}$ can be determined by $\sum_{ki} H_{ijkl} U_{ki} = \nabla_{lj}^{(t)}$. Solving this equation by directly inverting the huge tensor $H$ is however infeasible in practice and thus usually implemented by iterative methods such as

$$U_{ki}^{(v+1)} = \frac{\left( (A + D^B - B) U^{(v)} + \nabla^{(t)} \right)_{ki}}{D_{ii}^A}. \tag{18}$$

Such iterations albeit still form a costly inner loop over $v$. To overcome this, we initialize $U^{(0)} = 0$ and only employ the first iteration of each inner loop. Then one can find that such an approximated Newton's update rule $Y_{ki}^{(t+1)} = Y_{ki}^{(t)} - \frac{\nabla_{ki}^{(t)}}{D_{ii}^A}$ is identical to Equation (14). Such a first-step approximation technique has also been used in the Mean Shift algorithm as a generalized Expectation-Maximization solution [2].

## 5   A local mixture interpretation

Further rearranging the update rule can give us more insights of the properties of SSNE solutions:

$$Y_{ki}^{(t+1)} = \frac{\sum_j A_{ij} \left[ Y_{kj}^{(t)} + \frac{Q_{ij}}{P_{ij}} (Y_{ki}^{(t)} - Y_{kj}^{(t)}) \right]}{\sum_j A_{ij}}. \tag{19}$$

One can see that the above update rule mimics the maximization step in the EM-algorithm for classical Gaussian mixture model (e.g. [7]), or more particularly, the Mean Shift method [3, 2]. This resemblance inspires us to find an alternative interpretation of the SNE behavior in terms of a particular mixture model.

Given the current estimate $Y^{(t)}$, the fixed-point update rule actually performs minimization of

$$\sum_{ij} P_{ij} S_{ij} \| y_i - \mu_{ij}^{(t)} \|^2, \tag{20}$$

where $\mu_{ij}^{(t)} = y_j^{(t)} + \frac{Q_{ij}}{P_{ij}} \left( y_i^{(t)} - y_j^{(t)} \right)$. This problem is equivalent to maximizing the Jensen lower bound of

$$\log \sum_{ij} P_{ij} S_{ij} \exp \left( -\| y_i - \mu_{ij}^{(t)} \|^2 \right). \tag{21}$$

In this form, $\mu_{ij}^{(t)}$ can be regarded as the mean of the $j$-th mixture component for the $i$-th embedded data point, while the product $P_{ij} S_{ij}$ can be thought as the mixing coefficients[1]. Note that each data sample has its own mixing coefficients because of locality sensitivity.

For the converged estimate, i.e., $Y^{(t+1)} = Y^{(t)} = Y^*$, we can rewrite the mixture without the logarithm as

$$\sum_{ij} P_{ij} S_{ij} \exp \left\{ - \left( 1 - \frac{Q_{ij}}{P_{ij}} \right)^2 \| y_i^* - y_j^* \|^2 \right\}. \tag{22}$$

Maximizing this quantity clearly explains the ingredients of symmetric SNE: (1) $P_{ij}$ reflects that symmetric SNE favors close pairs in the input space, which is also adopted by most other locality

preserving methods. (2) As discussed in Section 3, $S_{ij}$ characterizes the tail heaviness of the embedding similarity function. For the baseline Gaussian similarity, this reduces to one and thus has no effect. For heavy-tailed similarities, $S_{ij}$ can compensate for mismatched dimensionalities between the input space and its embedding. (3) The first factor in the exponential emphasizes the distance graph matching, which underlies the success of SNE and its variants for capturing the global data structure compared with many other approaches that rely on only variance constraints [10]. A pair of $Q_{ij}$ that approximates $P_{ij}$ well can increase the exponential, while a pair with a poor mismatch yields little contribution to the mixture. (4) Finally, as credited in many other continuity preserving methods, the second factor in the exponential forces that close pairs in the input space are also situated nearby in the embedding space.

## 6 Experiments

### 6.1 t-SNE for unsupervised visualization

In this section we present experiments of unsupervised visualization with $T_1$ distribution, where our Fixed-Point t-SNE is compared with the original Gradient t-SNE optimization method as well as another dimensionality reduction approach, *Laplacian Eigenmap* [1]. Due to space limitation, we only focus on three data sets, *iris*, *wine*, and *segmentation* (training subset) from the UCI repository[2].

We followed the instructions in [10] for calculating $P_{ij}$ and choosing the learning rate $\eta$ and momentum amount $\beta(t)$ for Gradient t-SNE. Alternatively, we excluded two tricks, "early compression" and "early exaggeration", that are described in [10] from the comparison of long-run optimization because they apparently belong to the initialization stage. Here both Fixed-Point and Gradient t-SNEs execute with the same initialization which uses the "early compression" trick and pre-runs the Gradient t-SNE for 50 iterations as suggested in [10].

The visualization quality can be quantified using the ground truth class information. We adopt the measurement of the *homogeneity* of nearest neighbors:

$$\text{homogeneity} = \gamma/n, \tag{23}$$

where $\gamma$ is the number of mapped points belonging to the same class with their nearest neighbor and $n$ again is the total number of points. A larger homogeneity generally indicates better separability of the classes.

The experimental results are shown in Figure 2. Even though having a globally optimal solution, the Laplacian Eigenmap yields poor visualizations, since none of the classes can be isolated. By contrast, both t-SNE methods achieve much higher homogeneities and most clusters are well separated in the visualization plots. Comparing the two t-SNE implementations, one can see that our simple fixed-point algorithm converges even slightly faster than the comprehensive and carefully tuned Gradient t-SNE. Besides efficiency, our approach performs as good as Gradient t-SNE in terms of both t-SNE objectives and homogeneities of nearest neighbors for these data sets.

### 6.2 Semi-supervised visualization

Unsupervised symmetric SNE or t-SNE may perform poorly for some data sets in terms of identifying classes. In such cases it is better to include some supervised information and apply semi-supervised learning to enhance the visualization.

Let us consider another data set *vehicle* from the LIBSVM repository[3]. The top-left plot in Figure 3 demonstrates a poor visualization using unsupervised Gradient t-SNE. Next, suppose $10\%$ of the intra-class relationships are known. We can construct a supervised matrix $u$ where $u_{ij} = 1$ if $x_i$ and $x_j$ are known to belong to the same class and 0 otherwise. After normalizing $U_{ij} = u_{ij}/\sum_{a \neq b} u_{ab}$, we calculate the semi-supervised similarity matrix $\tilde{P} = (1-\rho)P+\rho U$, where the trade-off parameter $\rho$ is set to 0.5 in our experiments. All SNE learning algorithms remain unchanged except that $P$ is replaced with $\tilde{P}$.

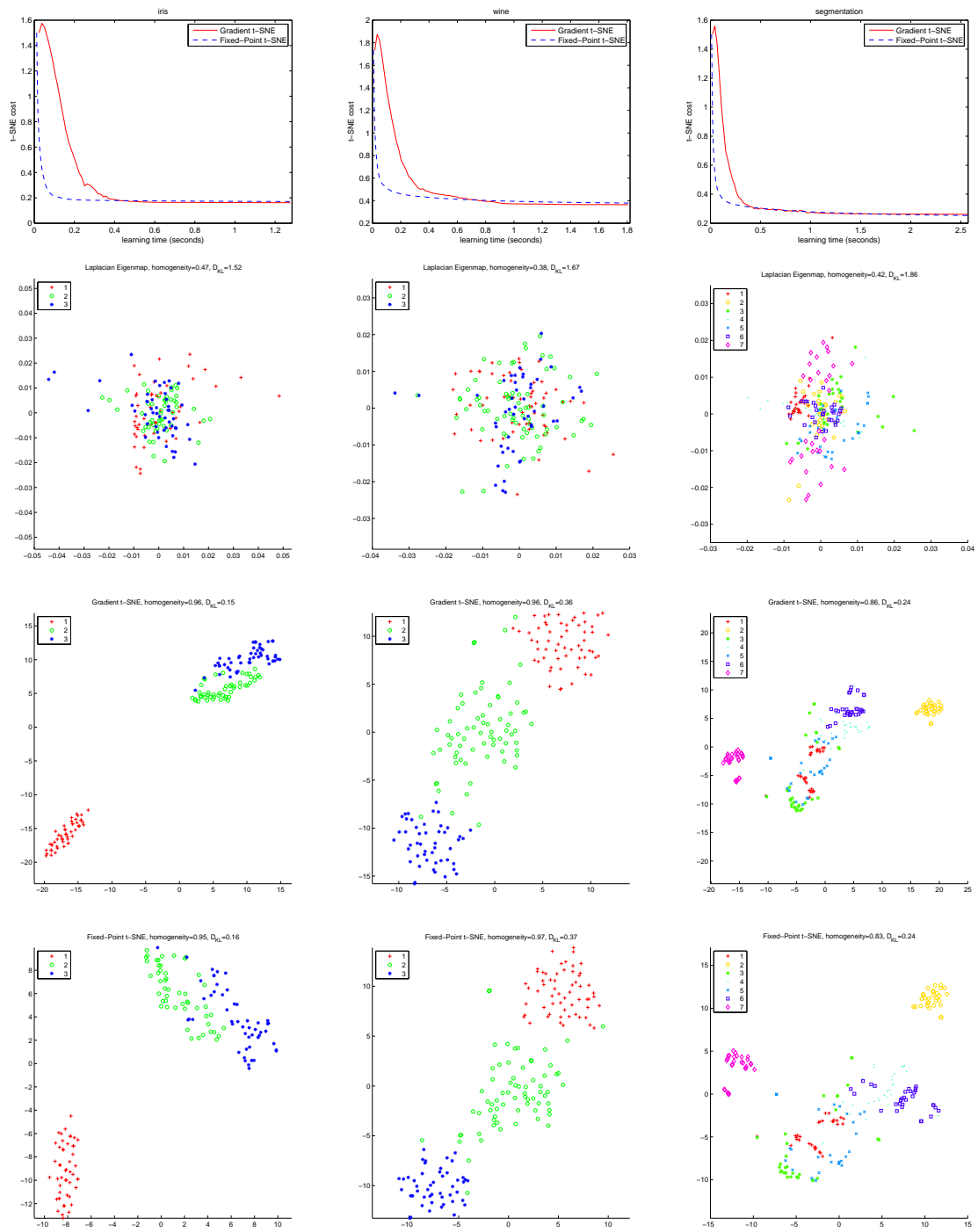

Figure 2: Unsupervised visualization on three data sets. Column 1 to 3 are results of *iris*, *wine* and *segmentation*, respectively. The first row comprises the learning times of Gradient and Fixed-Point t-SNEs. The second to fourth rows are visualizations using Laplacian Eigenmap, Gradient t-SNE, and Fixed-Point t-SNE, respectively.

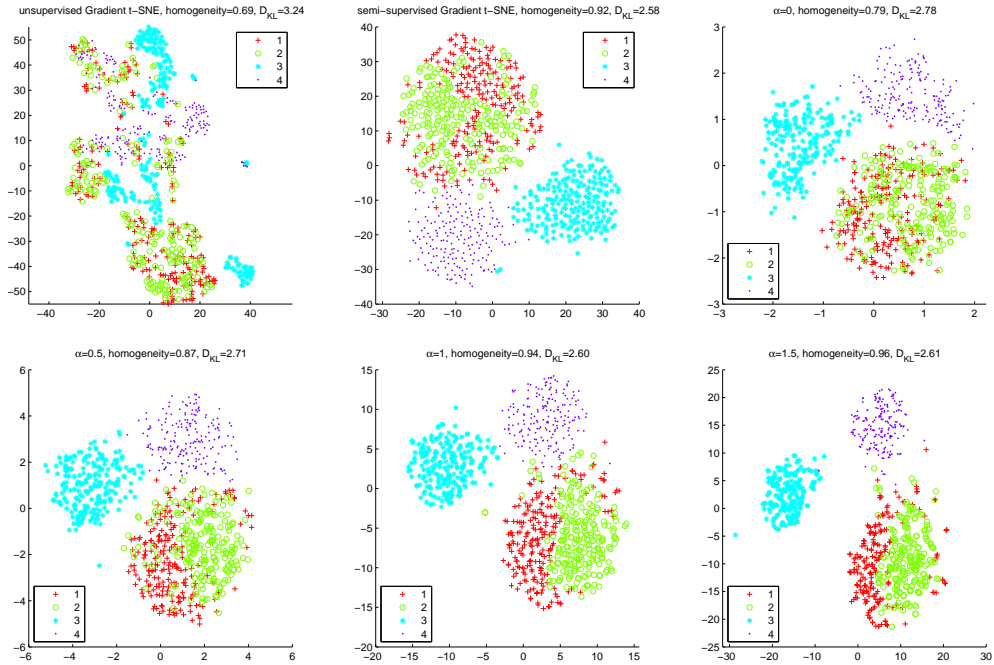

Figure 3: Semi-supervised visualization for the *vehicle* data set. The plots titled with $\alpha$ values are produced using the fixed-point algorithm of the power family of HSSNE.

The top-middle plot in Figure 3 shows that inclusion of some supervised information improves the homogeneity (0.92) and visualization, where Class 3 and 4 are identifiable, but the classes are still very close to each other, especially Class 1 and 2 heavily mixed. We then tried the power family of HSSNE with $\alpha$ ranging from 0 to 1.5, using our fixed-point algorithm. It can be seen that with $\alpha$ increased, the cyan and magenta clusters become more separate and Class 1 and 2 can also be identified. With $\alpha = 1$ and $\alpha = 2$, the HSSNEs implemented by our fixed-point algorithm achieve even higher homogeneities (0.94 and 0.96, respectively) than the Gradient t-SNE. On the other hand, too large $\alpha$ may increase the number of outliers and the Kullback-Leibler divergence.

# 7 Conclusions

The working mechanism of Heavy-tailed Symmetric Stochastic Neighbor Embedding (HSSNE) has been investigated rigorously. The several findings are: (1) we propose to use a negative score function to characterize and parameterize the heavy-tailed embedding similarity functions; (2) this finding has provided us with a power family of functions that convert distances to embedding similarities; and (3) we have developed a fixed-point algorithm for optimizing SSNE, which greatly saves the effort in tuning program parameters and facilitates the extensions and applications of heavy-tailed SSNE. We have compared HSSNE against t-SNE and Laplacian Eigenmap using UCI and LIBSVM repositories. Two sets of experimental results from unsupervised and semi-supervised visualization indicate that our method is efficient, accurate, and versatile over the other two approaches.

Our future work might include further empirical studies on the learning speed and robustness of HSSNE by using more extensive, especially large-scale, experiments. It also remains important to investigate acceleration techniques in both initialization and long-run stages of the learning.

# 8 Acknowledgement

The authors appreciate the reviewers for their extensive and informative comments for the improvement of this paper. This work is supported by a grant from the Research Grants Council of the Hong Kong Special Administrative Region, China (Project No. CUHK 4128/08E).

## Footnotes

[1]The data samples in such a symmetric mixture model do not follow the independent and identically distributed (i.i.d.) assumption because the mixing coefficient rows are not summed to the same number. Nevertheless, this does not affect our subsequent pairwise analysis.

[2]http://archive.ics.uci.edu/ml/

[3]http://www.csie.ntu.edu.tw/~cjlin/libsvmtools/datasets/

# References

[1] M. Belkin and P. Niyogi. Laplacian eigenmaps and spectral techniques for embedding and clustering. *Advances in neural information processing systems*, 14:585–591, 2002.

[2] M. A. Carreira-Perpiñán. Gaussian mean-shift is an em algorithm. *IEEE Transactions On Pattern Analysis And Machine Intelligence*, 29(5):767–776, 2007.

[3] D. Comaniciu and M. Peter. Mean Shift: A robust approach toward feature space analysis. *IEEE Transactions on Pattern Analysis and Machine Intelligence*, 24(5):603–619, 2002.

[4] J. A. Cook, I. Sutskever, A. Mnih, and G. E. Hinton. Visualizing similarity data with a mixture of maps. In *Proceedings of the 11th International Conference on Artificial Intelligence and Statistics*, volume 2, pages 67–74, 2007.

[5] M. Gashler, D. Ventura, and T. Martinez. Iterative non-linear dimensionality reduction with manifold sculpting. In J.C. Platt, D. Koller, Y. Singer, and S. Roweis, editors, *Advances in Neural Information Processing Systems 20*, pages 513–520. MIT Press, Cambridge, MA, 2008.

[6] G. Hinton and S. Roweis. Stochastic neighbor embedding. *Advances in Neural Information Processing Systems*, 15:833–840, 2003.

[7] G. J. McLachlan and D. Peel. *Finite Mixture Models*. Wiley, 2000.

[8] J. A. K. Suykens. Data visualization and dimensionality reduction using kernel maps with a reference point. *IEEE Transactions on Neural Networks*, 19(9):1501–1517, 2008.

[9] J. B. Tenenbaum, V. Silva, and J. C. Langford. A global geometric framework for nonlinear dimensionality reduction. *Science*, 290(5500):2319–2323, Dec. 2000.

[10] L. van der Maaten and G. Hinton. Visualizing data using t-SNE. *Journal of Machine Learning Research*, 9:2579–2605, 2008.
